# Polynomial Semantic Indexing

**Bing Bai**[(1)] **Jason Weston**[(1)(2)] **David Grangier**[(1)] **Ronan Collobert**[(1)]
**Kunihiko Sadamasa**[(1)] **Yanjun Qi**[(1)] **Corinna Cortes**[(2)] **Mehryar Mohri**[(2)(3)]

[(1)]NEC Labs America, Princeton, NJ
{bbai, dgrangier, collober, kunihiko, yanjun}@nec-labs.com
[(2)] Google Research, New York, NY
{jweston, corinna, mohri}@google.com
[(3)] NYU Courant Institute, New York, NY
mohri@cs.nyu.edu

## Abstract

We present a class of nonlinear (polynomial) models that are discriminatively trained to directly map from the word content in a query-document or document-document pair to a ranking score. Dealing with polynomial models on word features is computationally challenging. We propose a low-rank (but diagonal preserving) representation of our polynomial models to induce feasible memory and computation requirements. We provide an empirical study on retrieval tasks based on Wikipedia documents, where we obtain state-of-the-art performance while providing realistically scalable methods.

## 1   Introduction

Ranking text documents given a text-based query is one of the key tasks in information retrieval. A typical solution is to: (i) embed the problem in a feature space, e.g. model queries and target documents using a vector representation; and then (ii) choose (or learn) a similarity metric that operates in this vector space. Ranking is then performed by sorting the documents based on their similarity score with the query.

A classical vector space model, see e.g. [24], uses weighted word counts (e.g. via tf-idf) as the feature space, and the cosine similarity for ranking. In this case, the model is chosen by hand and *no machine learning* is involved. This type of model often performs remarkably well, but suffers from the fact that only exact matches of words between query and target texts contribute to the similarity score. That is, words are considered to be independent, which is clearly a false assumption.

Latent Semantic Indexing [8], and related methods such as pLSA and LDA [18, 2], are *unsupervised* methods that choose a low dimensional feature representation of "latent concepts" where words are no longer independent. They are trained with reconstruction objectives, either based on mean squared error (LSI) or likelihood (pLSA, LDA). These models, being unsupervised, are still agnostic to the particular task of interest.

More recently, supervised models for ranking texts have been proposed that can be trained on a *supervised* signal (i.e., labeled data) to provide a ranking of a database of documents given a query. For example, if one has click-through data yielding query-target relationships, one can use this to train these models to perform well on this task. Or, if one is interested in finding documents related to a given *query document*, one can use known hyperlinks to learn a model that performs well on this task. Many of these models have typically relied on optimizing over only a few hand-constructed features, e.g. based on existing vector space models such as tf-idf, the title, URL, PageRank and other information [20, 5]. In this work, we investigate an orthogonal research direction, as we analyze supervised methods that are based on *words* only. Such models are both more flexible, e.g. can be used for tasks such as cross-language retrieval, and can still be used in conjunction with

other features explored in previous work for further gains. At least one recent work, called Hash Kernels [25], has been proposed that does construct a word-feature based model in a learning-to-rank context.

In this article we define a class of nonlinear (polynomial) models that can capture higher order relationships between words. Our nonlinear representation of the *words* results in a very high dimensional feature space. To deal with this space we propose low rank (but diagonal preserving) representations of our polynomial models to induce feasible memory and computation requirements, resulting in a method that both exhibits strong performance and is tractable to train and test.

We show experimentally on retrieval tasks derived from Wikipedia that our method strongly outperforms other word based models, including tf-idf vector space models, LSI, query expansion, margin rank perceptrons and Hash Kernels.

The rest of this article is as follows. In Section 2, we describe our method, Section 3 discusses prior work, and Section 4 describes the experimental study of our method.

## 2 Polynomial Semantic Indexing

Let us denote the set of documents in the corpus as $\{d_t\}_{t=1}^{\ell} \subset \mathbb{R}^{\mathcal{D}}$ and a query text as $q \in \mathbb{R}^{\mathcal{D}}$, where $\mathcal{D}$ is the dictionary size, and the $j^{th}$ dimension of a vector indicates the frequency of occurrence of the $j^{th}$ word, e.g. using the tf-idf weighting and then normalizing to unit length.

Given a query $q$ and a document $d$ we wish to learn a (nonlinear) function $f(q, d)$ that returns a score measuring the relevance of $d$ given $q$. Let us first consider the naive approach of concatenating $(q, d)$ into a single vector and using $f(q, d) = w^{\top}[q, d]$ as a linear ranking model. This clearly does not learn anything useful as it would result in the same document ordering for any query, given fixed parameters $w$. However, considering a polynomial model:

$$f(q, d) = w^{\top} \Phi^k([q, d])$$

where $\Phi^k(\cdot)$ is a feature map that considers all possible $k$-degree terms:

$$\Phi^k(x_1, \ldots, x_{\mathcal{D}}) = \langle x_{i_1} \ldots x_{i_k} : 1 \leq i_1 \ldots i_k \leq \mathcal{D} \rangle$$

does render a useful discriminative model. For example for degree $k = 2$ we obtain:

$$f(q, d) = \sum_{ij} w_{ij}^1 q_i q_j + \sum_{ij} w_{ij}^2 d_i q_j + \sum_{ij} w_{ij}^3 d_i d_j$$

where $w$ has been rewritten as $w^1 \in \mathbb{R}^{\mathcal{D} \times \mathcal{D}}, w^2 \in \mathbb{R}^{\mathcal{D} \times \mathcal{D}}$ and $w^3 \in \mathbb{R}^{\mathcal{D} \times \mathcal{D}}$. The ranking order of documents $d$ given a fixed query $q$ is independent of $w^1$ and the value of the term with $w^3$ is independent of the query, so in the following we will consider models containing only terms with both $q$ and $d$. In particular, we will consider the following degree $k = 2$ model:

$$f^2(q, d) = \sum_{i,j=1}^{\mathcal{D}} W_{ij} q_i d_j = q^{\top} W d \tag{1}$$

where $W \in \mathbb{R}^{\mathcal{D} \times \mathcal{D}}$, and the degree $k = 3$ model:

$$f^3(q, d) = \sum_{i,j,k=1}^{\mathcal{D}} W_{ijk} q_i d_j d_k + f^2(q, d). \tag{2}$$

Note that if $W$ is an identity matrix in equation (1), we obtain the cosine similarity with tf-idf weighting. When other weights are nonzero this model can capture synonymy and polysemy as it looks at all possible cross terms, which can be tuned directly for the task of interest during training, e.g. the value of $W_{ij}$ corresponding to related words e.g. the word "jagger" in the query and "stones" in the target could be given a large value during training. The degree $k = 3$ model goes one stage further and can upweight $W_{ijk}$ for the triple "jagger", "stones" and "rolling" and can downweight the triple "jagger", "gem" and "stones". Note that we do not necessarily require preprocessing methods such as stemming here since these models can already match words with common stems (if

it is useful for the task). Note also that in equation (2) we could have just as easily have considered pairs of words in the query (rather than the document) as well.

Unfortunately, using such polynomial models is clearly infeasible for several reasons. Firstly, it will hardly be possible to fit $W$ in memory for realistic tasks. If the dictionary size is $\mathcal{D} = 30000$, then, for $k = 2$ this requires 3.4GB of RAM (assuming floats), and if the dictionary size is 2.5 Million (as it will be in our experiments in Section 4) this amounts to 14.5TB. For $k = 3$ this is even worse. Besides memory requirements, the huge number of parameters can of course also affect the generalization ability of this model.

We thus propose a low-rank (but diagonal preserving) approximation of these models which will lead to capacity control, faster computation speed and smaller memory footprint.

For $k = 2$ we propose to replace $W$ with $\overline{W}$, where

$$\overline{W}_{ij} = (U^\top V)_{ij} + I_{ij} = \sum_l U_{li} V_{lj} + I_{ij}.$$

Plugging this into equation (1) yields:

$$f_{LR}^2(q, d) = q^\top (U^\top V + I) d, \tag{3}$$

$$= \sum_{i=1}^N (Uq)_i (Vd)_i + q^\top d. \tag{4}$$

Here, $U$ and $V$ are $N \times \mathcal{D}$ matrices. Before looking at higher degree polynomials, let us first analyze this case. This induces a $N$-dimensional "latent concept" space in a way similar to LSI. However, this is different in several ways:

- First, and most importantly, we advocate training from a supervised signal using preference relations (ranking constraints).
- Further, $U$ and $V$ differ so it does not assume the query and target document should be embedded in the same way. This can hence model when the query text distribution is very different to the document text distribution, e.g. queries are often short and have different word occurrence and co-occurrence statistics. In the extreme case in cross language retrieval query and target texts are in different languages yet are naturally modeled in this setup.
- Finally, the addition of the identity term means this model automatically learns the trade-off between using the low dimensional space and a classical vector space model. This is important because the diagonal of the $W$ matrix gives the specificity of picking out when a word co-occurs in both documents (indeed, setting $W = I$ is equivalent to cosine similarity using tf-idf). The matrix $I$ is full rank and therefore cannot be approximated with the low-rank model $U^\top V$, so our model combines both terms in the approximation.

However, the efficiency and memory footprint are as favorable as LSI. Typically, one caches the $N$-dimensional representation for each document to use at query time.

For higher degree polynomials, e.g. $k = 3$ one can perform a similar approximation. Indeed, $W_{ijk}$ is approximated with

$$\overline{W}_{ijk} = \sum_l U_{li} V_{lj} Y_{lk}$$

where $U, V$ and $Y$ are $N \times \mathcal{D}$. When adding the diagonal preserving term and the lower order terms from the $k = 2$ polynomial, we obtain

$$f_{LR}^3(q, d) = \sum_{i=1}^N (Uq)_i (Vd)_i (Yd)_i + f_{LR}^2(q, d).$$

Clearly, we can approximate any degree $k$ polynomial using a product of $k$ linear embeddings in such a scheme. Note that at test time one can again cache the $N$-dimensional representation for each document by computing the product between the $V$ and $Y$ terms and are then still left with only $N$ multiplications per document for the embedding term at query time.

Interestingly, one can view this model as a "product of experts": the document is projected twice i.e. by two experts $V$ and $Y$ and the training will force them to focus on different aspects.

## 2.1 Training

Training such models could take many forms. In this paper we will adopt the typical "learning to rank" setup [17, 20]. Suppose we are given a set of tuples $\mathcal{R}$ (labeled data), where each tuple contains a query $q$, a relevant document $d^+$ and an non-relevant (or lower ranked) document $d^-$. We would like to choose $W$ such that $f(q, d^+) > f(q, d^-)$, that is $d^+$ should be ranked higher than $d^-$. We thus employ the margin ranking loss [17] which has already been used in several IR methods before [20, 5, 14], and minimize:

$$\sum_{(q,d^+,d^-)\in\mathcal{R}} \max(0, 1 - f(q, d^+) + f(q, d^-)). \qquad (5)$$

We train this using stochastic gradient descent, (see, e.g. [5]): iteratively, one picks a random tuple and makes a gradient step for that tuple. We choose the (fixed) learning rate which minimizes the training error. Convergence (or early stopping) is assessed with a validation set. Stochastic training is highly scalable and is easy to implement for our model. For example, for $k = 2$, one makes the following updates:

$$U \leftarrow U + \lambda V(d^+ - d^-)q^\top, \quad \text{if} \quad 1 - f_{LR}^2(q, d^+) + f_{LR}^2(q, d^-) > 0$$
$$V \leftarrow V + \lambda U q(d^+ - d^-)^\top, \quad \text{if} \quad 1 - f_{LR}^2(q, d^+) + f_{LR}^2(q, d^-) > 0.$$

Clearly, it is important to exploit the sparsity of $q$ and $d$ when calculating these updates. In our experiments we initialized the matrices $U$ and $V$ randomly using a normal distribution with mean zero and standard deviation one. The gradients for $k = 3$ are similar.

Note that researchers have also explored optimizing various alternative loss functions other than the ranking loss including optimizing normalized discounted cumulative gain (NDCG) and mean average precision (MAP) [5, 4, 6, 28]. In fact, one could use those optimization strategies to train our models instead of optimizing the ranking loss. One could also just as easily use them in unsupervised learning, such as in LSI, as well, e.g. by stochastic gradient descent on the reconstruction error.

## 3 Prior Work

Joachims et al. [20] trained a SVM with hand-designed features based on the title, body, search engines rankings and the URL. Burges et al. [5] proposed a neural network method using a similar set of features (569 in total). As described before, in contrast, we limited ourselves to body text (not using title, URL, etc.) and trained on millions of features based on these words.

The authors of [15] used a model similar to the naive full rank model (1), but for the task of image retrieval, and [13] also used a related (regression-based) method for advert placement. These techniques are implemented in related software to these two publications, PAMIR[1] and Vowpal Wabbit[2]. When the memory usage is too large, the latter bins the features randomly into a reduced space (hence with random collisions), a technique called Hash Kernels [25]. In all cases, the task of document retrieval, and the use of low-rank approximation or polynomial features is not studied. The current work generalizes and extends the Supervised Semantic Indexing approach [1] to general polynomial models.

Another related area of research is in distance metric learning [27, 19, 12]. Methods like LMNN [27] also learn a model similar to the naive full rank model (1), i.e. with the full matrix $W$ (but not with our improvements of this model that make it tractable for word features). They impose the constraint during the optimization that $W$ be a positive semidefinite matrix. Their method has considerable computational cost. For example, even after considerable optimization of the algorithm, it still takes 3.5 hours to train on 60,000 examples and 169 features (a pre-processed version of MNIST). This would hence not be scalable for large-scale text ranking experiments. Nevertheless, [7] compared LMNN [27], LEGO [19] and MCML [12] to a stochastic gradient method with a full matrix $W$ (identical to the model (1)) on a small image ranking task and reported in fact that the stochastic method provides both improved results and efficiency. Our method, on the other hand, both outperforms models like (1) and is feasible for word features, when (1) is not.

A tf-idf vector space model and LSI [8] are two standard baselines we will also compare to. We already mentioned pLSA [18] and LDA [2]; both have scalability problems and are not reported to generally outperform LSA and TF-IDF [11]. Query Expansion, often referred to as blind relevance feedback, is another way to deal with synonyms, but requires manual tuning and does not always yield a consistent improvement [29].

Several authors [23, 21] have proposed interesting nonlinear versions of *unsupervised* LSI using neural networks and showed they outperform LSI or pLSA. However, in the case of [23] we note their method is rather slow, and a dictionary size of only 2000 was used. A supervised method for LDA (sLDA) [3] has also been proposed where a set of auxiliary labels are trained on jointly with the unsupervised task. This provides supervision at the *document* level (via a class label or regression value) which is not a task of learning to rank, whereas here we study supervision at the (query,documents) level. The authors of [10] proposed "Explicit Semantic Analysis" which represents the meaning of texts in a high-dimensional space of concepts by building a feature space derived from the human-organized knowledge from an encyclopedia, e.g. Wikipedia. In the new space, cosine similarity is applied. Our method could be applied to such feature representations so that they are not agnostic to a particular supervised task as well.

As we will also evaluate our model over cross-language retrieval, we also briefly mention methods previously applied to this problem. These include first applying machine translation and then a conventional retrieval method such as LSI [16], a direct method of applying LSI for this task called CL-LSI [9], or using Kernel Canonical Correlation Analysis, KCCA [26]. While the latter is a strongly performing method, it also suffers from scalability problems.

## 4 Experimental Study

Learning a model of term correlations over a large vocabulary is a considerable challenge that requires a large amount of training data. Standard retrieval datasets like TREC[3] or LETOR [22] contain only a few hundred training queries, and are hence too small for that purpose. Moreover, some datasets only provide pre-processed features like tf, idf or BM25, and not the actual words. Click-through from web search engines could provide valuable supervision. However, such data is not publicly available.

We hence conducted experiments on Wikipedia and used links within Wikipedia to build a large-scale ranking task. We considered several tasks: document-document and query-document retrieval described in Section 4.1, and cross-language document-document retrieval described in Section 4.2.

In these experiments we compared our approach, Polynomial Semantic Indexing (PSI), to the following methods: tf-idf + cosine similarity (TFIDF), Query Expansion (QE), LSI[4], $\alpha$LSI + $(1 - \alpha)$ TFIDF, and the margin ranking perceptron and Hash Kernels with hash size $h$ using model (1). Query Expansion involves applying TFIDF and then adding mean vector $\beta \sum_{i=1}^{\mathcal{E}} d_{r_i}$ of the top $\mathcal{E}$ retrieved documents multiplied by a weighting $\beta$ to the query, and applying TFIDF again. For all methods, hyperparameters such as the embedding dimension $N \in \{50, 100, 200, 500, 1000\}$, $h \in \{1M, 3M, 6M\}$, $\alpha$, $\beta$ and $\mathcal{E}$ were chosen using a validation set.

For each method, we measured the ranking loss (the percentage of tuples in $\mathcal{R}$ that are incorrectly ordered), precision $P(n)$ at position $n = 10$ (P@10) and the mean average precision (MAP), as well as their standard deviations. For computational reasons, MAP and P@10 were measured by averaging over a fixed set of 1000 test queries, and the true test links and random subsets of 10,000 documents were used as the database, rather than the whole testing set. The ranking loss is measured using 100,000 testing tuples.

### 4.1 Document Retrieval

We considered a set of 1,828,645 English Wikipedia documents as a database, and split the 24,667,286 links randomly into two portions, 70% for training (plus validation) and 30% for test-

Table 1: Document-document ranking results on Wikipedia (limited dictionary size of 30,000 words). Polynomial Semantic Indexing (PSI) outperforms all baselines, and performs better with higher degree $k = 3$.

| Algorithm | Rank-Loss | MAP | P@10 |
|---|---|---|---|
| TFIDF | 1.62% | 0.329±0.010 | 0.163±0.006 |
| QE | 1.62% | 0.330±0.010 | 0.163±0.006 |
| LSI | 4.79% | 0.158±0.006 | 0.098±0.005 |
| $\alpha$LSI + $(1 - \alpha)$TFIDF | 1.28% | 0.346±0.011 | 0.170±0.007 |
| Margin Ranking Perceptron using (1) | 0.41% | 0.477±0.011 | 0.212±0.007 |
| PSI ($k = 2$) | 0.30% | 0.517±0.011 | 0.229±0.007 |
| PSI ($k = 3$) | **0.14%** | **0.539±0.011** | **0.236±0.007** |

Table 2: Empirical results for document-document ranking on Wikipedia (unlimited dictionary size).

| Algorithm | Rank-Loss | MAP | P@10 |
|---|---|---|---|
| TFIDF | 0.842% | 0.432±0.012 | 0.1933±0.007 |
| QE | 0.842% | 0.432±0.012 | 0.1933±0.007 |
| $\alpha$LSI + $(1 - \alpha)$TFIDF | 0.721% | 0.433±0.012 | 0.193±0.007 |
| Hash Kernels using (1) | 0.347% | 0.485±0.011 | 0.215±0.007 |
| PSI ($k = 2$) | 0.158% | 0.547±0.012 | 0.239±0.008 |
| PSI ($k = 3$) | **0.099%** | **0.590±0.012** | **0.249±0.008** |

Table 3: Empirical results for document-document ranking in two train/test setups: partitioning into train+test sets of links, or into train+test sets of documents with no cross-links (limited dictionary size of 30,000 words). The two setups yield. similar results.

| Algorithm | Testing Setup | Rank-Loss | MAP | P@10 |
|---|---|---|---|---|
| PSI ($k = 2$) | Partitioned links | 0.407% | 0.506±0.012 | 0.225±0.007 |
| PSI ($k = 2$) | Partitioned docs+links | 0.401% | 0.503±0.010 | 0.225±0.006 |

Table 4: Empirical results for query-document ranking on Wikipedia where query has $n$ keywords (this experiment uses a limited dictionary size of 30,000 words). For each $n$ we measure the ranking loss, MAP and P@10 metrics.

| Algorithm | $n = 5$ | | | $n = 10$ | | | $n = 20$ | | |
|---|---|---|---|---|---|---|---|---|---|
| | Rank | MAP | P@10 | Rank | MAP | P@10 | Rank | MAP | P@10 |
| TFIDF | 21.6% | 0.047 | 0.023 | 14.0% | 0.083 | 0.035 | 9.14% | 0.128 | 0.054 |
| $\alpha$LSI + $(1 - \alpha)$TFIDF | 14.2% | 0.049 | 0.023 | 9.73% | 0.089 | 0.037 | 6.36% | 0.133 | 0.059 |
| PSI ($k = 2$) | **4.37%** | **0.166** | **0.083** | **2.91%** | **0.229** | **0.100** | **1.80%** | **0.302** | **0.130** |

ing.[5] We then considered the following task: given a query document $q$, rank the other documents such that if $q$ links to $d$ then $d$ is highly ranked.

In our first experiment we constrained all methods to use only the top 30,000 most frequent words. This allowed us to compare to a margin ranking perceptron using model (1) which would otherwise not fit in memory. For our approach, Polynomial Semantic Indexing (PSI), we report results for degrees $k = 2$ and $k = 3$. Results on the test set are given in Table 1. Both variants of our method PSI strongly outperform the existing techniques. The margin rank perceptron using (1) can be seen as a full rank version of PSI for $k = 2$ (with $W$ unconstrained) but is outperformed by its low-rank counterpart – probably because it has too much capacity. Degree $k = 3$ outperforms $k = 2$, indicating that the higher order nonlinearities captured provide better ranking scores. For LSI and PSI embedding dimension $N = 200$ worked best, but other values gave similar results. In terms of other techniques, LSI is slightly better than TFIDF but QE in this case does not improve much over TFIDF, perhaps because of the difficulty of this task (there may too often be many irrelevant documents in the top $\mathcal{E}$ documents initially retrieved for QE to help).

Table 5: The closest five words in the document embedding space to some example query words.

| kitten | cat | cats | animals | species | dogs |
|---|---|---|---|---|---|
| vet | veterinarian | veterinary | medicine | animals | animal |
| ibm | computer | company | technology | software | data |
| nyc | york | new | manhattan | city | brooklyn |
| c++ | programming | windows | mac | unix | linux |
| xbox | console | game | games | microsoft | windows |
| beatles | mccartney | lennon | song | band | harrison |
| britney | spears | album | music | pop | her |

In our second experiment we no longer constrained methods to a fixed dictionary size, so all 2.5 million words are used. In this setting we compare to Hash Kernels which can deal with these dictionary sizes. The results, given in Table 2 show the same trends, indicating that the dictionary size restriction in the previous experiment did not bias the results in favor of any one algorithm. Note also that as a page has on average just over 3 test set links to other pages, the maximum P@10 one can achieve in this case is 0.31.

In some cases, one might be worried that our experimental setup has split training and testing data only by partitioning the links, but not the documents, hence performance of our model when new unseen documents are added to the database might be in question. We therefore also tested an experimental setup where the test set of documents is completely separate from the training set of documents, by completely removing all training set links between training and testing documents. In fact, this does not alter the performance significantly, as shown in Table 3.

**Query-Document Ranking**   So far, our evaluation uses whole Wikipedia articles as queries. One might wonder if the reported improvements also hold in a setup where queries consist of only a few keywords. We thus also tested our approach in this setup. We used the same setup as before but we constructed queries by keeping only $n$ random words from query documents in an attempt to mimic a "keyword search". Table 4 reports the results for keyword queries of length $n = 5, 10$ and 20. PSI yields similar improvements as in the document-document retrieval case over the baselines.

**Word Embedding**   The document embedding $Vd$ in equation (3) (similarly for the query embedding $Uq$) can be viewed as $Vd = \sum_i V_{\cdot i} d_i$, in which each column $V_{\cdot i}$ is the embedding of the word $d_i$. It is natural that semantically similar words are more likely to have similar embeddings. Table 5 shows a few examples. The first column contains query words, on the right are the 5 words with smallest Euclidean distance in the embedded space. We can see that they are quite relevant.

## 4.2   Cross Language Document Retrieval

Cross Language Retrieval [16] is the task of retrieving documents in a target language $E$ given a query in a different source language $F$. For example, Google provides such a service[6]. This is an interesting case for word-based learning to rank models which can naturally deal with this task without the need for machine translation as they directly learn the correspondence between the two languages from bi-lingual labeled data in the form of tuples $\mathcal{R}$. The use of a non-symmetric low-rank model like (3) also naturally suits this task (however in this case adding the identity does not make sense). We therefore also provide a case study in this setting.

We thus considered the same set of 1,828,645 English Wikipedia documents and a set of 846,582 Japanese Wikipedia documents, where 135,737 of the documents are known to be about the same concept as a corresponding English page (this information can be found in the wiki mark-up provided in a Wikipedia dump.) For example, the page about "Microsoft" can be found in both English and Japanese, and they are cross-referenced. These pairs are referred to as "mates" in the literature (see, e.g. [9]).

We then consider a cross language retrieval task that is analogous to the task in Section 4.1: given a Japanese query document $q_{Jap}$ that is the mate of the English document $q_{Eng}$, rank the English

Table 6: Cross-lingual Japanese document-English document ranking (limited dictionary size of 30,000 words).

| Algorithm | Rank-Loss | MAP | P@10 |
|---|---|---|---|
| $\text{TFIDF}_{EngEng}$(Google translated queries) | 4.78% | 0.319±0.009 | 0.259±0.008 |
| $\text{TFIDF}_{EngEng}$(ATLAS word-based translation) | 8.27% | 0.115±0.005 | 0.103±0.005 |
| $\text{TFIDF}_{EngEng}$(ATLAS translated queries) | 4.83% | 0.290±0.008 | 0.243±0.008 |
| $\text{LSI}_{EngEng}$(ATLAS translated queries) | 7.54% | 0.169±0.007 | 0.150±0.007 |
| $\alpha\text{LSI}_{EngEng}$(ATLAS)+$(1-\alpha)\text{TFIDF}_{EngEng}$(ATLAS) | 3.71% | 0.300±0.008 | 0.253±0.008 |
| $\text{CL-LSI}_{JapEng}$ | 9.29% | 0.190±0.007 | 0.161±0.007 |
| $\alpha\text{CL-LSI}_{JapEng}$+$(1-\alpha)\text{TFIDF}_{EngEng}$(ATLAS) | 3.31% | 0.275±0.009 | 0.212±0.008 |
| $\text{PSI}_{EngEng}$(ATLAS) | 1.72% | 0.399±0.009 | 0.325±0.009 |
| $\text{PSI}_{JapEng}$ | 0.96% | 0.438±0.009 | 0.351±0.009 |
| $\alpha\text{PSI}_{JapEng} + (1-\alpha)\text{TFIDF}_{EngEng}$(ATLAS) | 0.75% | 0.493±0.009 | 0.377±0.009 |
| $\alpha\text{PSI}_{JapEng} + (1-\alpha)\text{PSI}_{EngEng}$(ATLAS) | **0.63%** | **0.524±0.009** | **0.386±0.009** |

documents so that the documents linked to $q_{Eng}$ appear above the others. The document $q_{Eng}$ is removed and not considered during training or testing. The dataset is split into train/test as before.

The first type of baseline we considered is based on machine translation. We used a machine translation tool on the Japanese query, and then applied TFIDF or LSI. We considered three methods of machine translation: Google's API[7] or Fujitsu's ATLAS[8] was used to translate each query document, or we translated each word in the Japanese dictionary using ATLAS and then applied this word-based translation to a query. We also compared to CL-LSI [9] trained on all 90,000 Jap-Eng pairs from the training set.

For PSI, we considered two cases: (i) apply the ATLAS machine translation tool first, and then use PSI trained on the task in Section 4.1, e.g. the model given in equation (3) ($\text{PSI}_{EngEng}$), which was trained on English queries and English target documents; or (ii) train PSI directly with Japanese queries and English target documents using the model using (3) without the identity, which we call $\text{PSI}_{JapEng}$. We use degree $k = 2$ for PSI (trying $k = 3$ would have been interesting, but we have not performed this experiment). The results are given in Table 6. The dictionary size was again limited to the 30,000 most frequent words in both languages for ease of comparison with CL-LSI.

TFIDF using the three translation methods gave relatively similar results. Using LSI or CL-LSI slightly improved these results, depending on the metric. Machine translation followed by $\text{PSI}_{EngEng}$ outperformed all these methods, however the direct $\text{PSI}_{JapEng}$ which required no machine translation tool at all, improved results even further. We conjecture that this is because translation mistakes generate noisy features which $\text{PSI}_{JapEng}$ circumvents.

However, we also considered combining $\text{PSI}_{JapEng}$ with TFIDF or $\text{PSI}_{EngEng}$ using a mixing parameter $\alpha$ and this provided further gains at the expense of requiring a machine translation tool.

Note that many cross-lingual experiments, e.g. [9], typically measure the performance of finding a "mate", the same document in another language, whereas our experiment tries to model a query-based retrieval task. We also performed an experiment in the mate-finding setting. In this case, PSI achieves a ranking error of 0.53%, and CL-LSI achieves 0.81%.

## 5   Conclusion

We described a versatile, powerful set of discriminatively trained models for document ranking based on polynomial features over words, which was made feasible with a low-rank (but diagonal preserving) approximation. Many generalizations are possible: adding more features into our model, using other choices of loss function and exploring the use of the same models for tasks other than document retrieval, for example applying these models to ranking images rather than text, or to classification, rather than ranking, tasks.

## Footnotes

[1]http://www.idiap.ch/pamir/

[2]http://hunch.net/~vw/

[3]http://trec.nist.gov/

[4]We use the SVDLIBC software http://tedlab.mit.edu/~dr/svdlibc/ and the cosine distance in the latent concept space.

[5]We removed links to calendar years as they provide little information while being very frequent.

[6]http://translate.google.com/translate_s

[7]http://code.google.com/p/google-api-translate-java/

[8]http://www.fujitsu.com/global/services/software/translation/atlas/

# References

[1] B. Bai, J. Weston, R. Collobert, and D. Grangier. Supervised Semantic Indexing. In *European Conference on Information Retrieval*, pages 761–765, 2009.

[2] D. Blei, A. Ng, and M. Jordan. Latent dirichlet allocation. *Journal of Machine Learning Research*, 3:993–1022, 2003.

[3] D. M. Blei and J. D. McAuliffe. Supervised topic models. In *NIPS*, pages 121–128, 2007.

[4] C. Burges, R. Ragno, and Q. Le. Learning to Rank with Nonsmooth Cost Functions. In *NIPS*, pages 193–200, 2007.

[5] C. Burges, T. Shaked, E. Renshaw, A. Lazier, M. Deeds, N. Hamilton, and G. Hullender. Learning to Rank Using Gradient Descent. In *ICML*, pages 89–96, 2005.

[6] Z. Cao, T. Qin, T. Liu, M. Tsai, and H. Li. Learning to rank: from pairwise approach to listwise approach. In *ICML*, pages 129–136, 2007.

[7] G. Chechik, V. Sharma, U. Shalit, and S. Bengio. Large scale online learning of image similarity through ranking. In *(Snowbird) Learning Workshop*, 2009.

[8] S. Deerwester, S. Dumais, G. Furnas, T. Landauer, and R. Harshman. Indexing by latent semantic analysis. *JASIS*, 41(6):391–407, 1990.

[9] S. Dumais, T. Letsche, M. Littman, and T. Landauer. Automatic cross-language retrieval using latent semantic indexing. In *AAAI Spring Symposium on Cross-Language Text and Speech Retrieval*, pages 15–21, 1997.

[10] E. Gabrilovich and S. Markovitch. Computing semantic relatedness using wikipedia-based explicit semantic analysis. In *IJCAI*, pages 1606–1611, 2007.

[11] P. Gehler, A. Holub, and M. Welling. The rate adapting poisson (rap) model for information retrieval and object recognition. In *ICML*, pages 337–344, 2006.

[12] A. Globerson and S. Roweis. Visualizing pairwise similarity via semidefinite programming. In *AISTATS*, 2007.

[13] S. Goel, J. Langford, and A. Strehl. Predictive indexing for fast search. In *NIPS*, pages 505–512, 2008.

[14] D. Grangier and S. Bengio. Inferring document similarity from hyperlinks. In *CIKM*, pages 359–360, 2005.

[15] D. Grangier and S. Bengio. A discriminative kernel-based approach to rank images from text queries. *IEEE Transactions on Pattern Analysis and Machine Intelligence.*, 30(8):1371–1384, 2008.

[16] G. Grefenstette. *Cross-Language Information Retrieval*. Kluwer, Norwell, MA, USA, 1998.

[17] R. Herbrich, T. Graepel, and K. Obermayer. *Advances in Large Margin Classifiers*, chapter Large margin rank boundaries for ordinal regression. MIT Press, Cambridge, MA, 2000.

[18] T. Hofmann. Probabilistic latent semantic indexing. In *SIGIR*, pages 50–57, 1999.

[19] P. Jain, B. Kulis, I. S. Dhillon, and K. Grauman. Online metric learning and fast similarity search. In *NIPS*, pages 761–768, 2008.

[20] T. Joachims. Optimizing search engines using clickthrough data. In *SIGKDD*, pages 133–142, 2002.

[21] M. Keller and S. Bengio. A Neural Network for Text Representation. In *International Conference on Artificial Neural Networks*, 2005. IDIAP-RR 05-12.

[22] T. Liu, J. Xu, T. Qin, W. Xiong, and H. Li. Letor: Benchmark dataset for research on learning to rank for information retrieval. In *Proceedings of SIGIR 2007 Workshop on Learning to Rank*, 2007.

[23] R. Salakhutdinov and G. Hinton. Semantic Hashing. *Proceedings of the SIGIR Workshop on Information Retrieval and Applications of Graphical Models*, 2007.

[24] G. Salton and M. McGill. *Introduction to Modern Information Retrieval*. McGraw-Hill, 1986.

[25] Q. Shi, J. Petterson, G. Dror, J. Langford, A. Smola, A. Strehl, and V. Vishwanathan. Hash Kernels. In *AISTATS*, 2009.

[26] A. Vinokourov, J. Shawe-Taylor, and N. Cristianini. Inferring a Semantic Representation of Text via Cross-Language Correlation Analysis. In *NIPS*, pages 1497–1504, 2003.

[27] K. Weinberger and L. Saul. Fast solvers and efficient implementations for distance metric learning. In *ICML*, pages 1160–1167, 2008.

[28] Y. Yue, T. Finley, F. Radlinski, and T. Joachims. A support vector method for optimizing average precision. In *SIGIR*, pages 271–278, 2007.

[29] L. Zighelnic and O. Kurland. Query-drift prevention for robust query expansion. In *SIGIR*, pages 825–826, 2008.

